# Joint 3D Estimation of Objects and Scene Layout

**Andreas Geiger**
Karlsruhe Institute of Technology
geiger@kit.edu

**Christian Wojek**
MPI Saarbrücken
cwojek@mpi-inf.mpg.de

**Raquel Urtasun**
TTI Chicago
rurtasun@ttic.edu

## Abstract

We propose a novel generative model that is able to reason jointly about the 3D scene layout as well as the 3D location and orientation of objects in the scene. In particular, we infer the scene topology, geometry as well as traffic activities from a short video sequence acquired with a single camera mounted on a moving car. Our generative model takes advantage of dynamic information in the form of vehicle tracklets as well as static information coming from semantic labels and geometry (i.e., vanishing points). Experiments show that our approach outperforms a discriminative baseline based on multiple kernel learning (MKL) which has access to the same image information. Furthermore, as we reason about objects in 3D, we are able to significantly increase the performance of state-of-the-art object detectors in their ability to estimate object orientation.

## 1 Introduction

Visual 3D scene understanding is an important component in applications such as autonomous driving and robot navigation. Existing approaches produce either only qualitative results [11] or a mild level of understanding, e.g., semantic labels [10, 26], object detection [5] or rough 3D [15, 24]. A notable exception are approaches that try to infer the scene layout of indoor scenes in the form of 3D bounding boxes [13, 22]. However, these approaches can only cope with limited amounts of clutter (e.g., beds), and rely on the fact that indoor scenes satisfy very closely the manhattan world assumption, i.e., walls (and often objects) are aligned with the three dominant vanishing points. In contrast, outdoor scenarios often show more clutter, vanishing points are not necessarily orthogonal [25, 2], and objects often do not agree with the dominant vanishing points.

Prior work on 3D urban scene analysis is mostly limited to simple ground plane estimation [4, 29] or models for which the objects and the scene are inferred separately [6, 7]. In contrast, in this paper we propose a novel generative model that is able to reason jointly about the 3D scene layout as well as the 3D location and orientation of objects in the scene. In particular, given a video sequence of short duration acquired with a single camera mounted on a moving car, we estimate the scene topology and geometry, as well as the traffic activities and 3D objects present in the scene (see Fig. 1 for an illustration). Towards this goal we propose a novel image likelihood which takes advantage of dynamic information in the form of vehicle tracklets as well as static information coming from semantic labels and geometry (i.e., vanishing points). Interestingly, our inference reasons about whether vehicles are on the road, or parked, in order to get more accurate estimations. Furthermore, we propose a novel learning-based approach to detecting vanishing points and experimentally show improved performance in the presence of clutter when compared to existing approaches [19].

We focus our evaluation mainly on estimating the layout of intersections, as this is the most challenging inference task in urban scenes. Our approach proves superior to a discriminative baseline based on multiple kernel learning (MKL) which has access to the same image information (i.e., 3D tracklets, segmentation and vanishing points). We evaluate our method on a wide range of metrics including the accuracy of estimating the topology and geometry of the scene, as well as detecting

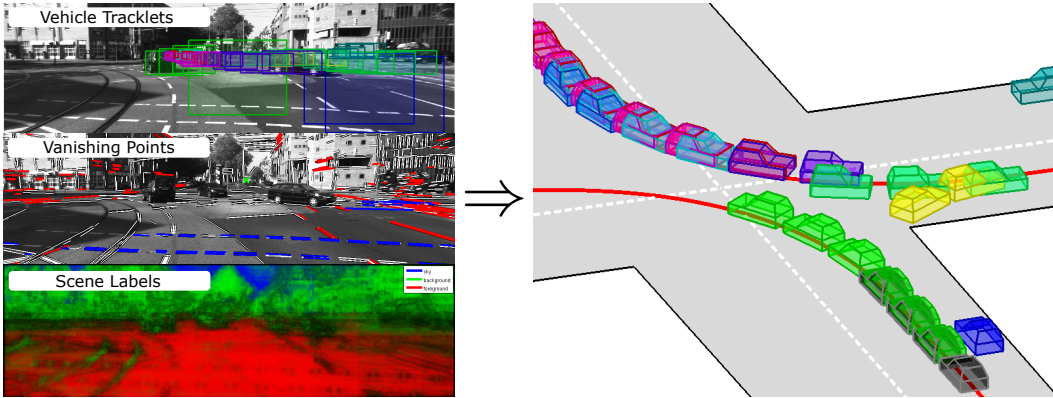

**Figure 1:** Monocular 3D Urban Scene Understanding. (Left) Image cues. (Right) Estimated layout: Detections belonging to a tracklet are depicted with the same color, traffic activities are depicted with red lines.

activities (i.e., traffic situations). Furthermore, we show that we are able to significantly increase the performance of state-of-the-art object detectors [5] in terms of estimating object orientation.

## 2 Related Work

While outdoor scenarios remain fairly unexplored, estimating the 3D layout of indoor scenes has experienced increased popularity in the past few years [13, 27, 22]. This can be mainly attributed to the success of novel structured prediction methods as well as the fact that indoor scenes behave mostly as "Manhattan worlds", i.e., edges on the image can be associated with parallel lines defined in terms of the three dominant vanishing points which are orthonormal. With a moderate degree of clutter, accurate geometry estimation has been shown for this scenario.

Unfortunately, most urban scenes violate the Manhattan world assumption. Several approaches have focused on estimating vanishing points in this more adversarial setting [25]. Barinova et al. [2] proposed to jointly perform line detection as well as vanishing point, azimut and zenith estimation. However, their approach does not tackle the problem of 3D scene understanding and 3D object detection. In contrast, we propose a generative model which jointly reasons about these two tasks.

Existing approaches to estimate 3D from single images in outdoor scenarios typically infer pop-ups [14, 24]. Geometric approaches, reminiscent to the blocksworld model, which impose physical constraints between objects (e.g., object A supports object B) have also been introduced [11]. Unfortunately, all these approaches are mainly qualitative and do not provide the level of accuracy necessary for real-world applications such as autonomous driving and robot navigation. Prior work on 3D traffic scene analysis is mostly limited to simple ground plane estimation [4], or models for which the objects and scene are inferred separately [6]. In contrast, our model offers a much richer scene description and reasons jointly about 3D objects and the scene layout.

Several methods have tried to infer the 3D locations of objects in outdoor scenarios [15, 1]. The most successful approaches use *tracklets* to prune spurious detections by linking consistent evidence in successive frames [18, 16]. However, these models are either designed for static camera setups in surveillance applications [16] or do not provide a rich scene description [18]. Notable exceptions are [3, 29] which jointly infer the camera pose and the location of objects. However, the employed scene models are rather simplistic containing only a single flat ground plane.

The closest approach to ours is probably the work of Geiger et al. [7], where a generative model is proposed in order to estimate the scene topology, geometry as well as traffic activities at intersections. Our work differs from theirs in two important aspects. First, they rely on stereo sequences while we make use of monocular imagery. This makes the inference problem much harder, as the noise in monocular imagery is strongly correlated with depth. Towards this goal we develop a richer image likelihood model that takes advantage of vehicle tracklets, vanishing points as well as segmentations of the scene into semantic labels. The second and most important difference is that Geiger et al. [7] estimate only the scene layout, while we reason jointly about the layout as well as the 3D location and orientation of objects in the scene (i.e., vehicles).

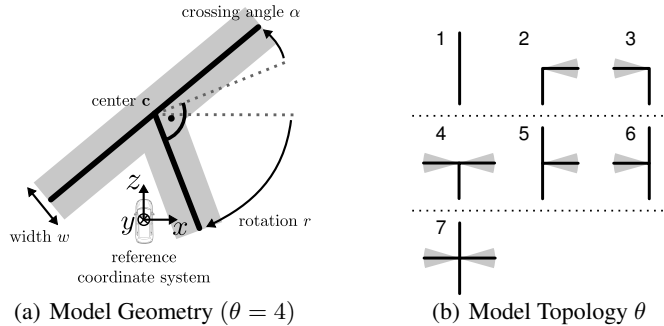

(a) Model Geometry ($\theta = 4$)  (b) Model Topology $\theta$

**Figure 2:** (a) Geometric model. In (b), the grey shaded areas illustrate the range of $\alpha$.

Finally, non-parametric models have been proposed to perform traffic scene analysis from a stationary camera with a view similar to bird's eye perspective [20, 28]. In our work we aim to infer similar activities but use video sequences from a camera mounted on a moving car with a substantially lower viewpoint. This makes the recognition task much more challenging. Furthermore, those models do not allow for viewpoint changes, while our model reasons about over 100 unseen scenes.

## 3 3D Urban Scene Understanding

We tackle the problem of estimating the 3D layout of urban scenes (i.e., road intersections) from monocular video sequences. In this paper 2D refers to observations in the image plane while 3D refers to the bird's eye perspective (in our scenario the height above ground is non-informative). We assume that the road surface is flat, and model the bird's eye perspective as the $y = 0$ plane of the standard camera coordinate system. The reference coordinate system is given by the position of the camera in the last frame of the sequence. The intrinsic parameters of the camera are obtained using camera calibration and the extrinsics using a standard Structure-from-Motion (SfM) pipeline [12].

We take advantage of dynamic and static information in the form of 3D vehicle tracklets, semantic labels (i.e., sky, background, road) and vanishing points. In order to compute 3D tracklets, we first detect vehicles in each frame independently using a semi-supervised version of the part-based detector of [5] in order to obtain orientation estimates. 2D tracklets are then estimated using 'tracking-by-detection': First adjacent frames are linked and then short tracklets are associated to create longer ones via the hungarian method. Finally, 3D vehicle tracklets are obtained by projecting the 2D tracklets into bird's eye perspective, employing error-propagation to obtain covariance estimates. This is illustrated in Fig. 1 where detections belonging to the same tracklet are grouped by color. The observer (i.e., our car) is shown in black. See sec 3.2 for more details on this process.

Since depth estimates in the monocular case are much noisier than in the stereo case, we employ a more constrained model than the one utilized in [7]. In particular, as depicted in Fig. 2, we model all intersection arms with the same width and force alternate arms to be collinear. We model lanes with splines (see red lines for active lanes in Fig. fig:motivation), and place parking spots at equidistant places along the street boundaries (see Fig. 3(b)). Our model then infers whether the cars participate in traffic or are parked in order to get more accurate layout estimations. Latent variables are employed to associate each detected vehicle with positions in one of these lanes or parking spaces. In the following, we first give an overview of our probabilistic model and then describe each part in detail.

### 3.1 Probabilistic Model

As illustrated in Fig. 2(b), we consider a fixed set of road layouts $\theta$, including straight roads, turns, 3- and 4- armed intersections. Each of these layouts is associated with a set of geometric random variables: The intersection center $\mathbf{c}$, the street width $w$, the global scene rotation $r$ and the angle of the crossing street $\alpha$ with respect to $r$ (see Fig. 2(a)). Note that for $\theta = 1$, $\alpha$ does not exist.

**Joint Distribution:** Our goal is to estimate the most likely configuration $\mathcal{R} = (\theta, \mathbf{c}, w, r, \alpha)$ given the image evidence $\mathcal{E} = \{\mathbf{T}, \mathbf{V}, \mathbf{S}\}$, which comprises *vehicle tracklets* $\mathbf{T} = \{\mathbf{t}_1, .., \mathbf{t}_N\}$, *vanish-*

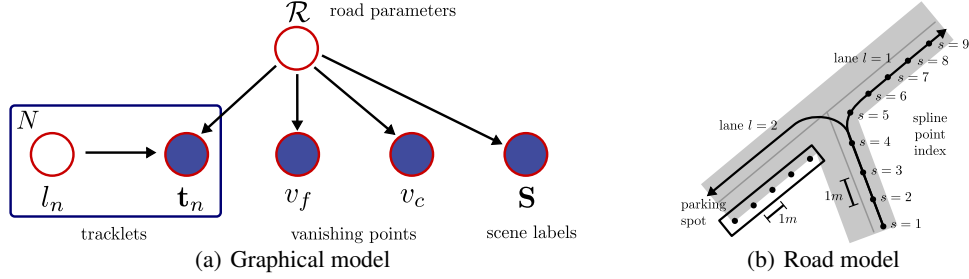

**Figure 3:** Graphical model and road model with lanes represented as B-splines.

*ing points* $\mathbf{V} = \{v_f, v_c\}$ and *semantic labels* $\mathbf{S}$. We assume that, given $\mathcal{R}$, all observations are independent. Fig. 3(a) depicts our graphical model which factorizes the joint distribution as

$$p(\mathcal{E}, \mathcal{R}|\mathcal{C}) = p(\mathcal{R}) \underbrace{\left[\prod_{n=1}^{N} \sum_{l_n} p(\mathbf{t}_n, l_n|\mathcal{R}, \mathcal{C})\right]}_{\text{Vehicle Tracklets}} \underbrace{p(v_f|\mathcal{R}, \mathcal{C})p(v_c|\mathcal{R}, \mathcal{C})}_{\text{Vanishing Points}} \underbrace{p(\mathbf{S}|\mathcal{R}, \mathcal{C})}_{\text{Semantic Labels}} \quad (1)$$

where $\mathcal{C}$ are the (known) extrinsic and intrinsic camera parameters for all the frames in the video sequence, $N$ is the total number of tracklets and $\{l_n\}$ denotes latent variables representing the lane or parking positions associated with every vehicle tracklet. See Fig. 3(b) for an illustration.

**Prior:** Let us first define a scene prior, which factorizes as

$$p(\mathcal{R}) = p(\theta)p(\mathbf{c}, w)p(r)p(\alpha) \quad (2)$$

where $\mathbf{c}$ and $w$ are modeled jointly to capture their correlation. We model $w$ using a log-Normal distribution since it takes only positive values. Further, since it is highly multimodal, we model $p(\alpha)$ in a non-parametric fashion using kernel density estimation (KDE), and define:

$$r \sim \mathcal{N}(\mu_r, \sigma_r) \qquad (\mathbf{c}, \log w)^T \sim \mathcal{N}(\boldsymbol{\mu}_{cw}, \boldsymbol{\Sigma}_{cw}) \qquad \theta \sim \delta(\theta_{MAP})$$

In order to avoid the requirement for trans-dimensional inference procedures, the topology $\theta_{MAP}$ is estimated a priori using joint boosting, and set fixed at inference. To estimate $\theta_{MAP}$, we use the same feature set employed by the MKL baseline (see Sec. 4 for details).

## 3.2 Image Likelihood

This section details our image likelihood for tracklets, vanishing points and semantic labels.

**Vehicle Tracklets:** In the following, we drop the tracklet index $n$ to simplify notation. Let us define a 3D tracklet as a set of object detections $\mathbf{t} = \{\mathbf{d}_1, .., \mathbf{d}_M\}$. Here, each object detection $\mathbf{d}_m = (f_m, \mathbf{b}_m, \mathbf{o}_m)$ contains the frame index $f_m \in \mathbb{N}$, the object bounding box $\mathbf{b}_m \in \mathbb{R}^4$ defined as 2D position and size, as well as a normalized orientation histogram $\mathbf{o}_m \in \mathbb{R}^8$ with 8 bins. We compute the bounding box $\mathbf{b}_m$ and orientation $\mathbf{o}_m$ by supervised training of a part-based object detector [5], where each component contains examples from a single orientation. Following [5], we apply the softmax function on the output scores and associate frames using the hungarian algorithm in order to obtain tracklets.

As illustrated in Fig. 3(b), we represent drivable locations with splines, which connect incoming and outgoing lanes of the intersection. We also allow cars to be parked on the side of the road, see Fig. 3(b) for an illustration. Thus, for a $K$-armed intersection, we have $l \in \{1, .., K(K-1) + 2K\}$ in total, where $K(K-1)$ is the number of lanes and $2K$ is the number of parking areas. We use the latent variable $l$ to index the lane or parking position associated with a tracklet. The joint probability of a tracklet $\mathbf{t}$ and its lane index $l$ is given by $p(\mathbf{t}, l|\mathcal{R}, \mathcal{C}) = p(\mathbf{t}|l, \mathcal{R}, \mathcal{C})p(l)$. We assume a uniform prior over lanes and parking positions $l \sim \mathcal{U}(1, K(K-1) + 2K)$, and denote the posterior by $p_l$ when $l$ corresponds to a lane, and $p_p$ when it is a parking position.

In order to evaluate the tracklet posterior for lanes $p_l(\mathbf{t}|l, \mathcal{R}, \mathcal{C})$, we need to associate all object detections $\mathbf{t} = \{\mathbf{d}_1, .., \mathbf{d}_M\}$ to locations on the spline. We do this by augmenting the observation

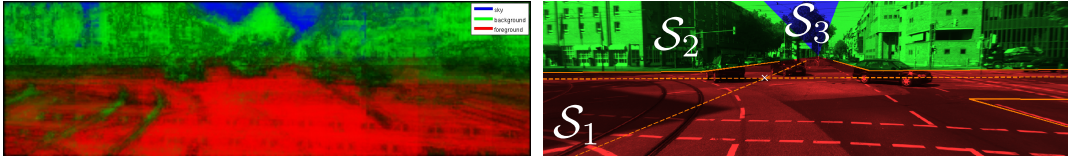

**Figure 4: Scene Labels:** Scene labels obtained from joint boosting (left) and from our model (right).

model with an additional latent variable $s$ per object detection $\mathbf{d}$ as illustrated in Fig. 3(b). The posterior is modeled using a left-to-right Hidden Markov Model (HMM), defined as:

$$p_l(\mathbf{t}|l,\mathcal{R},\mathcal{C}) = \sum_{s_1,..,s_M} p_l(s_1)p_l(\mathbf{d}_1|s_1,l,\mathcal{R},\mathcal{C}) \prod_{m=2}^{M} p_l(s_m|s_{m-1})p_l(\mathbf{d}_m|s_m,l,\mathcal{R},\mathcal{C}) \quad (3)$$

We constrain all tracklets to move forward in 3D by defining the transition probability $p(s_m|s_{m-1})$ as uniform on $s_m \geq s_{m-1}$ and 0 otherwise. Further, uniform initial probabilites $p_l(s_1)$ are employed, since no location information is available a priori. We assume that the emission likelihood $p_l(\mathbf{d}_m|s_m,l,\mathcal{R},\mathcal{C})$ factorizes into the object location and its orientation. We impose a multinomial distribution over the orientation $p_l(f_m,\mathbf{o}_m|s_m,l,\mathcal{R},\mathcal{C})$, where each object orientation votes for its bin as well as neighboring bins, accounting for the uncertainty of the object detector. The 3D object location is modeled as a Gaussian with uniform outlier probability $c_l$

$$p_l(f_m,\mathbf{b}_m|s_m,l,\mathcal{R},\mathcal{C}) \propto c_l + \mathcal{N}(\boldsymbol{\pi}_m|\boldsymbol{\mu}_m,\boldsymbol{\Sigma}_m) \quad (4)$$

where $\boldsymbol{\pi}_m = \boldsymbol{\pi}_m(f_m,\mathbf{b}_m,\mathcal{C}) \in \mathbb{R}^2$ denotes the object detection mapped into bird's eye perspective, $\boldsymbol{\mu}_m = \boldsymbol{\mu}_m(s_m,l,\mathcal{R}) \in \mathbb{R}^2$ is the coordinate of the spline point $s_m$ on lane $l$ and $\boldsymbol{\Sigma}_m = \boldsymbol{\Sigma}_m(f_m,\mathbf{b}_m,\mathcal{C}) \in \mathbb{R}^{2\times 2}$ is the covariance of the object location in bird's eye coordinates.

We now describe how we transform the 2D tracklets into 3D tracklets $\{\boldsymbol{\pi}_1,\boldsymbol{\Sigma}_1,..,\boldsymbol{\pi}_M,\boldsymbol{\Sigma}_M\}$, which we use in $p_l(\mathbf{d}_m|s_m,l,\mathcal{R},\mathcal{C})$: We project the image coordinates into bird's eye perspective by back-projecting objects into 3D using several complementary cues. Towards this goal we use the 2D bounding box foot-point in combination with the estimated road plane. Assuming typical vehicle dimensions obtained from annotated ground truth, we also exploit the width and height of the bounding box. Covariances in bird's eye perspective are obtained by error-propagation. In order to reduce noise in the observations we employ a Kalman smoother with constant 3D velocity model.

Our parking posterior model is similar to the lane posterior described above, except that we do not allow parked vehicles to move; We assume them to have arbitrary orientations and place them at the sides of the road. Hence, we have

$$p_p(\mathbf{t}|l,\mathcal{R},\mathcal{C}) = \sum_{s} \prod_{m=1}^{M} p_p(\mathbf{d}_m|s,l,\mathcal{R},\mathcal{C})p(s) \quad (5)$$

with $s$ the index for the parking spot location within a parking area and

$$p_p(\mathbf{d}_m|s,l,\mathcal{R},\mathcal{C}) = p_p(f_m,\mathbf{b}_m|s,l,\mathcal{R},\mathcal{C}) \propto c_p + \mathcal{N}(\boldsymbol{\pi}_m|\boldsymbol{\mu}_m,\boldsymbol{\Sigma}_m) \quad (6)$$

Here, $c_p$, $\boldsymbol{\pi}_m$ and $\boldsymbol{\Sigma}_m$ are defined as above, while $\boldsymbol{\mu}_m = \boldsymbol{\mu}_m(s,l,\mathcal{R}) \in \mathbb{R}^2$ is the coordinate of the parking spot location in bird's eye perspective (see Fig. 3(b) for an illustration). For inference, we subsample each tracklet trajectory equidistantly in intervals of 5 meters in order to reduce the number of detections within a tracklet and keep the total evaluation time of $p(\mathcal{R},\mathcal{E}|\mathcal{C})$ low.

**Vanishing Points:** We detect two types of dominant vanishing points (VP) in the last frame of each sequence: $v_f$ corresponding to the forward facing street and $v_c$ corresponding to the crossing street. While $v_f$ is usually in the image, the $u$-coordinate of the crossing VP is often close to infinity (see Fig. 1). As a consequence, we represent $v_f \in \mathbb{R}$ by its image $u$-coordinate and $v_c \in [-\frac{\pi}{4},\frac{\pi}{4}]$ by the angle of the crossing road, back projected into the image.

Following [19], we employ a line detector to reason about dominant VPs in the scene. We relax the original model of [19] to allow for non-orthogonal VPs, as intersection arms are often non-orthogonal. Unfortunately, traditional VP detectors tend to fail in the presence of clutter, which our images exhibit to a large extent, for example generated by shadows. To tackle this problem we

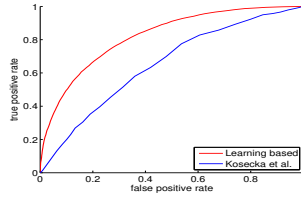

| | Error |
|---|---|
| Felzenszwalb et al. [5] (raw) | 32.6 ° |
| Felzenszwalb et al. [5] (smoothed) | 31.2 ° |
| Our method ($\theta$ unknown) | **15.7** ° |
| Our method ($\theta$ known) | **13.7** ° |

(a) Detecting Structured Lines  (b) Object Orientation Error

**Figure 5: Detecting Structured Lines and Object Orientation Errors:** Our approach outperforms [19] in the task of VP estimation, and [5] in estimating the orientation of objects.

reweight line segments according to their likelihood of carrying structural information. To this end, we learn a k-nn classifier on an annotated training database where lines are labeled as either *structure* or *clutter*. Here, *structure* refers to line segments that are aligned with the major orientations of the road, as well as facade edges of buildings belonging to dominant VPs. Our feature set comprises geometric information in the form of position, length, orientation and number of lines with the same orientation as well as perpendicular orientation in a local window. The local appearance is represented by the mean, standard deviation and entropy of all pixels on both sides of the line. Finally, we add texton-like features using a Gabor filter bank, as well as 3 principal components of the scene GIST [23]. The *structure* k-nn classifier's confidence is used in the VP voting process to reweight the lines. The benefit of our learning-based approach is illustrated in Fig. 5.

To avoid estimates from spurious outliers we threshold the dominant VPs and only retain the most confident ones. We assume that $v_f$ and $v_c$ are independent given the road parameters. Let $\mu_f = \mu_f(\mathcal{R}, \mathcal{C})$ be the image $u$-coordinate (in pixels) of the forward facing street's VP and let $\mu_c = \mu_c(\mathcal{R}, \mathcal{C})$ be the orientation (in radians) of the crossing street in the image. We define

$$p(v_f|\mathcal{R},\mathcal{C}) \quad \propto c_f + \delta_f \, \mathcal{N}(v_f|\mu_f, \sigma_f) \qquad p(v_c|\mathcal{R},\mathcal{C}) \quad \propto c_c + \delta_c \, \mathcal{N}(v_c|\mu_c, \sigma_c)$$

where $\{c_f, c_c\}$ are small constants capturing outliers, $\{\delta_f, \delta_c\}$ take value 1 if the corresponding VP has been detected in the image and 0 otherwise, and $\{\sigma_f, \sigma_c\}$ are parameters of the VP model.

**Semantic Labels:** We segment the last frame of the sequence pixelwise into 3 semantic classes, i.e., *road*, *sky* and *background*. For each patch, we infer a score for each of the 3 labels using the boosting algorithm of [30] with a combination of Walsh-Hadamard filters [30], as well as multi-scale features developed for detecting man-made structures [21] on patches of size $16 \times 16$, $32 \times 32$ and $64 \times 64$. We include the latter ones as they help in discriminating buildings from road. For training, we use a set of 200 hand-labeled images which are not part of the test data.

Given the softmax normalized label scores $S_{u,v}^{(i)} \in \mathbb{R}$ of each class $i$ for the patch located at position $(u, v)$ in the image, we define the likelihood of a scene labeling $\mathbf{S} = \{\mathbf{S}^{(1)}, \mathbf{S}^{(2)}, \mathbf{S}^{(3)}\}$ as

$$p(\mathbf{S}|\mathcal{R},\mathcal{C}) \propto \exp(\gamma \sum_{i=1}^{3} \sum_{(u,v) \in \mathcal{S}_i} S_{u,v}^{(i)}) \tag{7}$$

where $\gamma$ is a model parameter and $\mathcal{S}_i$ is the set of all pixels of class $i$ obtained from the reprojection of the geometric model into the image. Note that the road boundaries directly define the lower end of a facade while we assume a typical building height of 4 stories, leading to the upper end. Facades adjacent to the observers own' street are not considered. Fig. 4 illustrates an example of the scene labeling returned by boosting (left) as well as the labeling generated from the reprojection of our model (right). Note that a large overlap corresponds to a large likelihood in Eq. 7

### 3.3 Learning and Inference

Our goal is to estimate the posterior of $\mathcal{R}$, given the image evidence $\mathcal{E}$ and the camera calibration $\mathcal{C}$:

$$p(\mathcal{R}|\mathcal{E},\mathcal{C}) \propto p(\mathcal{E}|\mathcal{R},\mathcal{C})p(\mathcal{R}) \tag{8}$$

**Learning the prior:** We estimate the parameters of the prior $p(\mathcal{R})$ using maximum likelihood leave-one-out cross-validation on the scene database of [7]. This is straightforward as the prior in Eq. 2 factorizes. We employ KDE with $\sigma = 0.02$ to model $p(\alpha)$, as it works well in practice.

|                           |          | Location | Orientation | Overlap | Activity |
|---------------------------|----------|----------|-------------|---------|----------|
| (Inference with known $\theta$) | Baseline | 6.0 m    | 9.6 deg     | 44.9 %  | 18.4 %   |
|                           | Ours     | **5.8 m** | **5.9 deg** | **53.0 %** | **11.5** % |

|                           |          | $\theta$ | Location | Orientation | Overlap | Activity |
|---------------------------|----------|----------|----------|-------------|---------|----------|
| (Inference with unknown $\theta$) | Baseline | 27.4 %  | **6.2 m** | 21.7 deg    | 39.3 %  | 28.1 %   |
|                           | Ours     | **70.8** % | 6.6 m  | **7.2 deg** | **48.1 %** | **16.6 %** |

**Figure 6: Inference of topology and geometry** .

|        | k       | Location | Orientation | Overlap | Activity |
|--------|---------|----------|-------------|---------|----------|
| Stereo | 92.9 %  | 4.4 m    | 6.6 deg     | 62.7 %  | 8.0 %    |
| Ours   | 71.7 %  | 6.6 m    | 7.2 deg     | 48.1 %  | 16.6 %   |

**Figure 7: Comparison with stereo** when $k$ and $\theta$ are unknown.

**Learning the 3D tracklet parameters:** Eq. 4 requires a function $\varphi : f, \mathbf{b}, \mathcal{C} \rightarrow \boldsymbol{\pi}, \boldsymbol{\Sigma}$ which takes a frame index $f \in \mathbb{N}$, an object bounding box $\mathbf{b} \in \mathbb{R}^4$ and the calibration parameters $\mathcal{C}$ as input and maps them to the object location $\boldsymbol{\pi} \in \mathbb{R}^2$ and uncertainty $\boldsymbol{\Sigma} \in \mathbb{R}^{2 \times 2}$ in bird's eye perspective. As cues for this mapping we use the bounding box width and height, as well as the location of the bounding box foot-point. Scene depth adaptive error propagation is employed for obtaining $\boldsymbol{\Sigma}$. The unknown parameters of the mapping are the uncertainty in bounding box location $\sigma_u, \sigma_v$, width $\sigma_{\Delta u}$ and height $\sigma_{\Delta v}$ as well as the real-world object dimensions $\Delta_x, \Delta_y$ along with their uncertainties $\sigma_{\Delta x}, \sigma_{\Delta y}$. We learn these parameters using a separate training dataset, including 1020 images with 3634 manually labeled vehicles and depth information [8].

**Inference:** Since the posterior in Eq. 8 cannot be computed in closed form, we approximate it using Metropolis-Hastings sampling [9]. We exploit a combination of *local* and *global* moves to obtain a well-mixing Markov chain. While local moves modify $\mathcal{R}$ slightly, global moves sample $\mathcal{R}$ directly from the prior. This ensures quickly traversing the search space, while still exploring local modes. To avoid trans-dimensional jumps, the road layout $\theta$ is estimated separately beforehand using MAP estimation $\theta_{MAP}$ provided by joint boosting [30]. We pick each of the remaining elements of $\mathcal{R}$ at random and select local and global moves with equal probability.

## 4 Experimental Evaluation

In this section, we first show that learning which line features convey structural information improves dominant vanishing point detection. Next, we compare our method to a multiple kernel learning (MKL) baseline in estimating scene topology, geometry and traffic activities on the dataset of [7], but only employing information from a single camera. Finally, we show that our model can significantly improve object orientation estimates compared to state-of-the-art part based models [5]. For all experiments, we set $c_l = c_p = 10^{-15}$, $\sigma_f = 0.1$, $c_f = 10^{-10}$, $\sigma_c = 0.01$, $c_c = 10^{-30}$ and $\gamma = 0.1$.

**Vanishing Point Estimation:** We use a database of 185 manually annotated images to learn a predictor of which line segments are *structured*. This is important since cast shadows often mislead the VP estimation process. Fig. 5(a) shows the ROC curves for the method of [19] relaxed to non-orthogonal VPs (blue) as well as our learning-based approach (red). While the baseline gets easily disturbed by clutter, our method is more accurate and has significantly less false positives.

**3D Urban Scene Inference:** We evaluate our method's ability to infer the scene layout by building a competitive baseline based on multi-kernel Gaussian process regression [17]. We employ a total of 4 kernels built on GIST [23], tracklet histograms, VPs as well as scene labels. Note that these are the same features employed by our model to estimate the scene topology, $\theta_{MAP}$. For the tracklets, we discretize the $50 \times 50$ m area in front of the vehicle into bins of size $5 \times 5$ m. Each bin consists of four binary elements, indicating whether forward, backward, left or right motion has been observed at that location. The VPs are included with their value as well as an indicator variable denoting whether the VP has been found or not. For each semantic class, we compute histograms at 3 scales, which divide the image into $3 \times 1$, $6 \times 2$ and $12 \times 4$ bins, and concatenate them. Following [7] we measure error in terms of the location of the intersection center in meters, the orientation of the intersection arms in degrees, the overlap of road area with ground truth as well as the percentage of correctly discovered intersection crossing activities. For details about these metrics we refer the reader to [7].

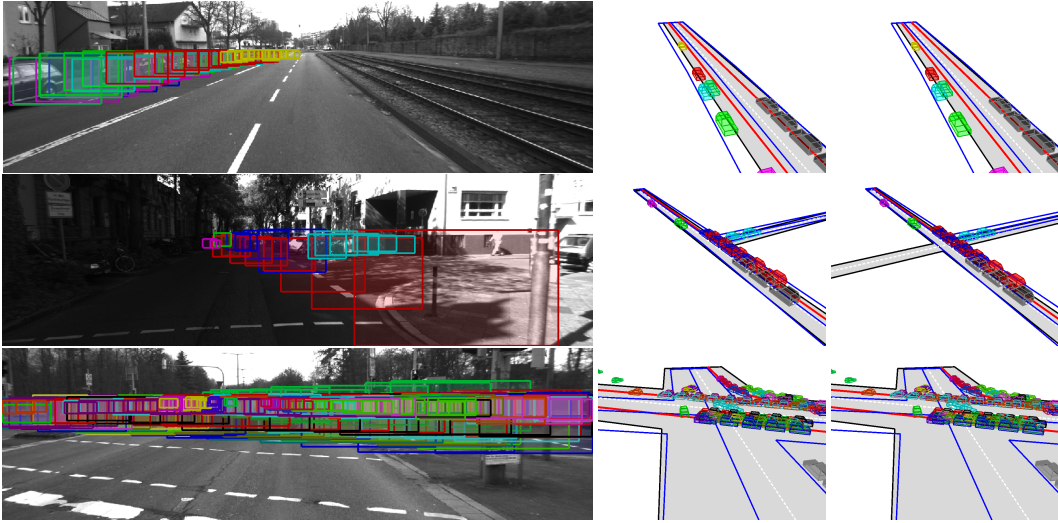

**Figure 8: Automatically inferred scene descriptions.** (Left) Trackets from all frames superimposed. (Middle) Inference result with $\theta$ known and (Right) $\theta$ unknown. The inferred intersection layout is shown in gray, ground truth labels are given in blue. Detected activities are marked by red lines.

We perform two types of experiments: In the first one we assume that the type of intersection $\theta$ is given, and in the second one we estimate $\theta$ as well. As shown in Fig. 6, our method significantly outperforms the MKL baseline in almost all error measures. Our method particularly excels in estimating the intersection arm orientations and activities. We also compare our approach to [7] in Fig. 7. As this approach uses stereo cameras, it can be considered as an oracle, yielding the highest performance achievable. Our approach is close to the oracle; The difference in performance is due to the depth uncertainties that arise in the monocular case, which makes the problem much more ambiguous. Fig. 8 shows qualitative results, with detections belonging to the same tracklet depicted with the same color. The trajectories of all the trackets are superimposed in the last frame. Note that, while for the 2-armed and 4-armed case the topology has been estimated correctly, the 3-armed case has been confused with a 4-armed intersection. This is our most typical failure mode. Despite this, the orientations are correctly estimated and the vehicles are placed at the correct locations.

**Improving Object Orientation Estimation:** We also evaluate the performance of our method in estimating 360 degree object orientations. As cars are mostly aligned with the road surface, we only focus on the orientation angle in bird's eye coordinates. As a baseline, we employ the part-based detector of [5] trained in a supervised fashion to distinguish between 8 canonical views, where each view is a mixture component. We correct for the ego motion and project the highest scoring orientation into bird's eye perspective. For our method, we infer the scene layout $\mathcal{R}$ using our approach and associate every tracklet to its lane by maximizing $p_l(l|\mathbf{t}, \mathcal{R}, \mathcal{C})$ over $l$ using Viterbi decoding. We then select the tangent angle at the associated spline's footpoint $s$ on the inferred lane $l$ as our orientation estimate. Since parked cars are often oriented arbitrarily, our evaluation focuses on moving vehicles only. Fig. 5(b) shows that we are able to significantly reduce the orientation error with respect to [5]. This also holds true for the smoothed version of [5], where we average orientations over temporally neighboring bins within each tracklet.

## 5   Conclusions

We have proposed a generative model which is able to perform joint 3D inference over the scene layout as well as the location and orientation of objects. Our approach is able to infer the scene topology and geometry, as well as traffic activities from a short video sequence acquired with a single camera mounted on a car driving around a mid-size city. Our generative model proves superior to a discriminative approach based on MKL. Furthermore, our approach is able to outperform significantly a state-of-the-art detector on its ability to estimate 3D object orientation. In the future, we plan to incorporate more discriminative cues to further boost performance in the monocular case. We also believe that incorporating traffic sign states and pedestrians into our model will be an interesting avenue for future research towards fully understanding complex urban scenarios.

# References

[1] S. Bao, M. Sun, and S. Savarese. Toward coherent object detection and scene layout understanding. In *CVPR*, 2010.

[2] O. Barinova, V. Lempitsky, E. Tretyak, and P. Kohli. Geometric image parsing in man-made environments. In *ECCV*, 2010.

[3] W. Choi and S. Savarese. Multiple target tracking in world coordinate with single, minimally calibrated camera. In *ECCV*, 2010.

[4] A. Ess, B. Leibe, K. Schindler, and L. Van Gool. Robust multi-person tracking from a mobile platform. *PAMI*, 31:1831–1846, 2009.

[5] P. Felzenszwalb, R.Girshick, D. McAllester, and D. Ramanan. Object detection with discriminatively trained part-based models. *PAMI*, 32:1627–1645, 2010.

[6] D. Gavrila and S. Munder. Multi-cue pedestrian detection and tracking from a moving vehicle. *IJCV*, 73:41–59, 2007.

[7] A. Geiger, M. Lauer, and R. Urtasun. A generative model for 3d urban scene understanding from movable platforms. In *Computer Vision and Pattern Recognition*, 2011.

[8] A. Geiger, M. Roser, and R. Urtasun. Efficient large-scale stereo matching. In *Asian Conference on Computer Vision*, 2010.

[9] W. Gilks and S. Richardson, editors. *Markov Chain Monte Carlo in Practice*. Chapman & Hall, 1995.

[10] S. Gould, T. Gao, and D. Koller. Region-based segmentation and object detection. In *NIPS*, 2009.

[11] A. Gupta, A. Efros, and M. Hebert. Blocks world revisited: Image understanding using qualitative geometry and mechanics. In *ECCV*, 2010.

[12] R. Hartley and A. Zisserman. *Multiple View Geometry in Computer Vision*. Cambridge, 2004.

[13] V. Hedau, D. Hoiem, and D.A. Forsyth. Recovering the spatial layout of cluttered rooms. In *ICCV*, 2009.

[14] D. Hoiem, A. Efros, and M. Hebert. Recovering surface layout from an image. *IJCV*, 75:151–172, 2007.

[15] D. Hoiem, A. Efros, and M. Hebert. Putting objects in perspective. *IJCV*, 80:3–15, 2008.

[16] C. Huang, B. Wu, and R. Nevatia. Robust object tracking by hierarchical association of detection responses. In *ECCV*, 2008.

[17] A. Kapoor, K. Grauman, R. Urtasun, and T. Darrell. Gaussian processes for object categorization. *IJCV*, 88:169–188, 2010.

[18] R. Kaucic, A. Perera, G. Brooksby, J. Kaufhold, and A. Hoogs. A unified framework for tracking through occlusions and across sensor gaps. In *CVPR*, 2005.

[19] J. Kosecka and W. Zhang. Video compass. In *ECCV*, 2002.

[20] D. Kuettel, M. Breitenstein, L. Gool, and V. Ferrari. What's going on?: Discovering spatio-temporal dependencies in dynamic scenes. In *CVPR*, 2010.

[21] S. Kumar and M. Hebert. Man-made structure detection in natural images using a causal multiscale random field. In *CVPR*, 2003.

[22] D. Lee, A. Gupta, M. Hebert, and T. Kanade. Estimating spatial layout of rooms using volumetric reasoning about objects and surfaces. In *NIPS*, 2010.

[23] A. Oliva and A. Torralba. Modeling the shape of the scene: a holistic representation of the spatial envelope. *IJCV*, 42:145–175, 2001.

[24] A. Saxena, S. H. Chung, and A. Y. Ng. 3-D depth reconstruction from a single still image. *IJCV*, 76:53–69, 2008.

[25] G. Schindler and F. Dellaert. Atlanta world: An expectation maximization framework for simultaneous low-level edge grouping and camera calibration in complex man-made environments. In *CVPR*, 2004.

[26] J. Shotton, J. Winn, C. Rother, and A. Criminisi. Textonboost for image understanding: Multi-class object recognition and segmentation by jointly modeling texture, layout, and context. *IJCV*, 81:2–23, 2009.

[27] H. Wang, S. Gould, and D. Koller. Discriminative learning with latent variables for cluttered indoor scene understanding. In *ECCV*, 2010.

[28] X. Wang, X. Ma, and W. Grimson. Unsupervised activity perception in crowded and complicated scenes using hierarchical bayesian models. *PAMI*, 2009.

[29] C. Wojek, S. Roth, K. Schindler, and B. Schiele. Monocular 3D Scene Modeling and Inference: Understanding Multi-Object Traffic Scenes. In *ECCV*, 2010.

[30] C. Wojek and B. Schiele. A dynamic CRF model for joint labeling of object and scene classes. In *ECCV*, 2008.

